# "FAST LEARNING IN MULTI-RESOLUTION HIERARCHIES"

John Moody

Yale Computer Science, P.O. Box 2158, New Haven, CT 06520

### Abstract

A class of fast, supervised learning algorithms is presented. They use local representations, hashing, and multiple scales of resolution to approximate functions which are piece-wise continuous. Inspired by Albus's CMAC model, the algorithms learn orders of magnitude more rapidly than typical implementations of back propagation, while often achieving comparable qualities of generalization. Furthermore, unlike most traditional function approximation methods, the algorithms are well suited for use in real time adaptive signal processing. Unlike simpler adaptive systems, such as linear predictive coding, the adaptive linear combiner, and the Kalman filter, the new algorithms are capable of efficiently capturing the structure of complicated non-linear systems. As an illustration, the algorithm is applied to the prediction of a chaotic timeseries.

## 1 Introduction

A variety of approaches to adaptive information processing have been developed by workers in disparate disciplines. These include the large body of literature on approximation and interpolation techniques (curve and surface fitting), the linear, real-time adaptive signal processing systems (such as the adaptive linear combiner and the Kalman filter), and most recently, the reincarnation of non-linear neural network models such as the multilayer perceptron.

Each of these methods has its strengths and weaknesses. The curve and surface fitting techniques are excellent for off-line data analysis, but are typically not formulated with real-time applications in mind. The linear techniques of adaptive signal processing and adaptive control are well-characterized, but are limited to applications for which linear descriptions are appropriate. Finally, neural network learning models such as back propagation have proven extremely versatile at learning a wide variety of non-linear mappings, but tend to be very slow computationally and are not yet well characterized.

The purpose of this paper is to present a general description of a class of supervised learning algorithms which combine the ability of the conventional curve

fitting and multilayer perceptron methods to precisely learn non-linear mappings with the speed and flexibility required for real-time adaptive application domains.

The algorithms are inspired by a simple, but often overlooked, neural network model, Albus's Cerebellar Model Articulation Controller (CMAC) [2,1], and have a great deal in common with the standard techniques of interpolation and approximation. The algorithms "learn from examples", generalize well, and can perform efficiently in real time. Furthermore, they overcome the problems of precision and generalization which limit the standard CMAC model, while retaining the CMAC's speed.

## 2    System Description

The systems are designed to rapidly approximate mappings $g : \vec{x} \mapsto \vec{y}$ from multi-dimensional input spaces $\vec{x} \in \mathcal{S}_{input}$ to multidimensional output spaces $\vec{y} \in \mathcal{S}_{output}$. The algorithms can be applied to any problem domain for which a metric can be defined on the input space (typically the Euclidean, Hamming, or Manhattan metric) and for which the desired learned mapping is (to a close approximation) piece-wise continuous. (Discontinuities in the desired mapping, such as those at classification boundaries, are approximated continuously.) Important general classes of such problems include approximation of real-valued functions $\mathcal{R}^n \mapsto \mathcal{R}^m$ (such as those found in signal processing), classification problems $\mathcal{R}^n \mapsto \mathcal{B}^m$ (such as phoneme classification), and boolean mapping problems $\mathcal{B}^n \mapsto \mathcal{B}^m$ (such as the NETtalk problem [20]). Here, $\mathcal{R}$ are the reals and $\mathcal{B}$ is $\{0,1\}$. This paper focuses on real-valued mappings; the formulation and application of the algorithms to boolean problem domains will be presented elsewhere.

In order to specify the complete learning system in detail, it is easiest to start with simple special cases and build the description from the bottom up.

### 2.1    A Simple Adaptive Module

The simplest special case of the general class under consideration is described as follows. The input space is overlayed with a lattice of points $\vec{x}^\beta$ a local function value or "weight" $\vec{V}^\beta$ is assigned to every possible lattice point. The output of the system for a given input is:

$$\vec{z}(\vec{x}) = \left[ \sum_\beta \vec{V}^\beta N^\beta(\vec{x}) \right] , \tag{1}$$

where $N^\beta(\vec{x})$ is a neighborhood function for the $\beta^{th}$ lattice point such that $N^\beta = 1$ if $\vec{x}^\beta$ is the lattice point closest to the input vector $\vec{x}$ and $N^\beta = 0$ otherwise.

More generally, the neighborhood functions $N$ can overlap and the sum in equation (1) can be replaced by an average. This results in a greater ability to generalize when training data is sparse, but at the cost of losing fine detail.

Learning is accomplished by varying the $\vec{V}^\beta$ to minimize the squared error of the system output on a set of training data:

$$E = \frac{1}{2} \sum_i (\vec{z}_i^{desired} - \vec{z}(\vec{x}_i))^2 \;, \tag{2}$$

where the sum is over all exemplars $\{\, \vec{x}_i,\; \vec{z}_i^{desired}\,\}$ in the training set. The determination of $\vec{V}^\beta$ is easily formulated as a real time adaptive algorithm by using gradient descent to minimize an instantaneous estimate $E(t)$ of the error:

$$\frac{dV}{dt} = -\eta \frac{dE(t)}{dV} \;. \tag{3}$$

## 2.2 Saving Memory with Hashing: The CMAC

The approach of the previous section encounters serious difficulty when the dimension of the input space becomes large and the distribution of data in the input space becomes highly non-uniform. In such cases, allocating a separate function value for each possible lattice point is extremely wasteful, because the majority of lattice points will have no training data within a local neighborhood.

As an example, suppose that the input space is four dimensional, but that all input data lies on a fuzzy two dimensional subspace. (Such a situation [projected onto 3-dimensions] is shown in figure [2A].) Furthermore, suppose that the input space is overlayed with a rectangular lattice with $K$ nodes per dimension. The complete lattice will contain $K^4$ nodes, but only $O(K^2)$ of those nodes will have training data in their local neighborhoods Thus, only $O(K^2)$ of the weights $V^\beta$ will have any meaning. The remaining $O(K^4)$ weights will be wasted. (This assumes that the lattice is not too fine. If $K$ is too large, then only $O(P)$ of the lattice points will have training data nearby, where $P$ is the number of training data.)

An alternative approach is to have only a small number of weights and to allocate them to only those regions of the input space which are populated with training data. This allocation can be accomplished by a dimensionality-reducing mapping from a virtual lattice in the input space onto a lookup table of weights or function values. In the absence of any *a priori* information about the distribution of data in the input space, the optimal mapping is a random mapping, for example a universal hashing function [8]. The random nature of such a function insures that neighborhood relationships in the virtual lattice are not preserved. The average behavior of an ensemble of universal hashing functions is thus to access all elements of the lookup table with equal probability, regardless of the correlations in the input data.

The many-to-one hash function can be represented here as a matrix $H^{\tau\beta}$ of 0's and 1's with one 1 per column, but many 1's per row. With this notation, the system response function is:

$$\vec{z}(\vec{x}) = \sum_{\tau=1}^{T} \sum_{\beta=1}^{N} \vec{V}^\tau H^{\tau\beta} N^\beta(\vec{x}) \;. \tag{4}$$

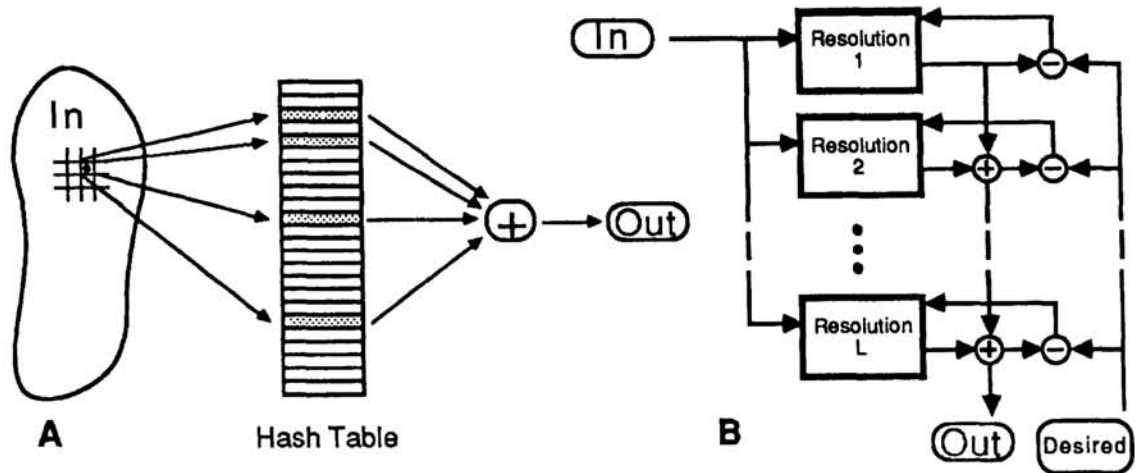

Figure 1: (A) A simple CMAC module. (B) The computation of errors for a multi-resolution hierarchy.

The CMAC model of Albus is obtained when a distributed representation of the input space is used and the neighborhood functions $N^\beta(\vec{x})$ are overlapping. In this case, the sum over $\beta$ is replaced by an average. Note that, as specified by equation (4), hash table collisions are not resolved. This introduces "collision noise", but the effect of this noise is reduced by $1/\sqrt{(B)}$, where $B$ is the number of neighborhood functions which respond to a given input. Collision noise can be completely eliminated if standard collision resolution techniques are used.

A few comments should be made about efficiency. In spite of the costly formal sums in equation (4), actual implementations of the algorithm are extremely fast. The set of non-zero $N^\beta(\vec{x})$ on the virtual lattice, the hash function value for each vertex, and the set of corresponding lookup table values $\vec{V}^\tau$ given by the hash function are easily determined on the fly. The entire hash function $H^{\tau\beta}$ is never pre-computed, the sum over the index $\beta$ is limited to a few lattice points neighboring the input $\vec{x}$, and since each lattice point is associated with only one lookup table value, the formal sum over $\tau$ disappears.

The CMAC model is shown schematically in figure [1A].

## 2.3  Interpolation: Neighborhood Functions with Graded Response

One serious problem with the formulations discussed so far is that the neighborhood functions are constant in their regions of support. Thus, the system response is discontinuous over neighborhood boundaries. This problem can be easily remedied by using neighborhood functions with graded response in order to perform continuous interpolation between lattice points.

The normalized system response function is then:

$$\vec{z}(\vec{x}) = \left[\sum_\tau \sum_\beta \vec{V}^\tau H^{\tau\beta} R^\beta(\vec{x})\right] / \left[\sum_\beta R^\beta(\vec{x})\right] . \tag{5}$$

The functions $R^\beta(\vec{x})$ are the graded neighborhood response functions associated with each lattice point $\vec{x}^\beta$. They are intended to have local support on the input space $S_{input}$, thus being non-zero only in a local neighborhood of their associated lattice point $\vec{x}^\beta$. Each function $R^\beta(\vec{x})$ attains its maximum value at lattice point $\vec{x}^\beta$ and drops off monotonically to zero as the distance $\|\vec{x}^\beta - \vec{x}\|$ increases. Note that $R$ is not necessarily isotropic or symmetric.

Certain classes of localized response functions $R$ defined on certain lattices are self-normalized, meaning that:

$$\sum_\beta R^\beta(\vec{x}) = 1 , \text{ for any } \vec{x} . \tag{6}$$

In this case, the equation (5) simplifies to:

$$\vec{z}(\vec{x}) = \sum_\tau \sum_\beta \vec{V}^\tau H^{\tau\beta} R^\beta(\vec{x}) \tag{7}$$

One particularly important and useful class of of response functions are the $B$-splines. However, it is not easy to formulate B-splines on arbitrary lattices in high dimensional spaces.

## 2.4 Multi-Resolution Interpolation

The final limitation of the methods described so far is that they use a lattice at only one scale of resolution. Without detailed *a priori* knowledge of the distribution of data in the input space, it is difficult to choose an optimal lattice spacing. Furthermore, there is almost always a trade-off between the ability to generalize and the ability to capture fine detail. When a single coarse resolution is used, generalization is good, but fine details are lost. When a single fine resolution is used, fine details are captured in those regions which contain dense data, but no general picture emerges for those regions in which data is sparse.

Good generalization and fine detail can both be captured by using a multi-resolution hierarchy.

A hierarchical system with $L$ levels represents functions $g : \vec{x} \mapsto \vec{y}$ in the following way:

$$\vec{y}(\vec{x}) \equiv \vec{y}_L(\vec{x}) = \sum_{\lambda=1}^L \vec{z}_\lambda(\vec{x}) , \tag{8}$$

where $\vec{z}_\lambda$ is a mapping as described in equation(5) for the $\lambda$-th level in the hierarchy. The coarsest scale is $\lambda = 1$ and the finest is $\lambda = L$.

The multi-resolution system is trained such that the finer scales learn corrections to the total output of the coarser scales. This is accomplished by using a hierarchy of error functions. For each level in the hierarchy $\lambda$, the output for that level $\vec{y}_\lambda$ is defined to be the partial sum

$$\vec{y}_\lambda = \sum_{\kappa=1}^{\lambda} z_\kappa \ .$$

(Note that $\vec{y}_{\lambda+1} = \vec{y}_\lambda + \vec{z}_{\lambda+1}$.) The error for level $\lambda$ is defined to be

$$E_\lambda = \sum_i E_\lambda(i) \ ,$$

where the error associated with the $i^{th}$ exemplar is:

$$E_\lambda(i) = \frac{1}{2}(\vec{y}_i^{des} - \vec{y}_\lambda(\vec{x}_i))^2 \ .$$

The learning or training procedure for level $\lambda$ involves varying the lookup table values $V_\lambda^r$ for that level to minimize $E_\lambda$. Note that the lookup table values $V_\kappa^r$ for previous or subsequent levels ($\kappa \neq \lambda$) are held fixed during the minimization of $E_\lambda$. Thus, the lookup table values for each level are varied to minimize *only* the error defined for that level. This hierarchical learning procedure guarantees that the first level mapping $z_1$ is the best possible at that level, the second level mapping $z_2$ constitutes the best possible corrections to the first level, and the $\lambda$-th level mapping $z_\lambda$ constitutes the best possible corrections to the total contributions of all previous levels. The computation of error signals is shown schematically in figure [1B].

It should be noted that multi-resolution approaches have been successfully used in other contexts. Examples are the well-known multigrid methods for solving differential equations and the pyramid architectures used in machine vision [6,7].

# 3   Application to Timeseries Prediction

The multi-resolution hierarchy can be applied to a wide variety of problem domains as mentioned earlier. Due to space limitations, we consider only one test problem here, the prediction of a chaotic timeseries.

As it is usually formulated, the prediction is accomplished by finding a real-valued mapping $f : \mathcal{R}^n \mapsto \mathcal{R}$ which takes a sequence of $n$ recent samples of the timeseries and predicts the value at a future moment. Typically, the state space imbedding in $\mathcal{R}^n$ is $\vec{x}[t] = (x[t], x[t-\Delta], x[t-2\Delta], x[t-3\Delta])$, where $\Delta$ is the sampling parameter, and the correct prediction for prediction time $T$ is $x[t+T]$. For the purposes of testing various non-parametric prediction methods, it is assumed that the underlying process which generates the timeseries is unknown.

The particular timeseries studied here results from integrating the Mackey-Glass differential-delay equation [14]:

$$\frac{dx[t]}{dt} = -b\,x[t] + a\,\frac{x[t-\tau]}{1+x[t-\tau]^{10}} \ . \tag{9}$$

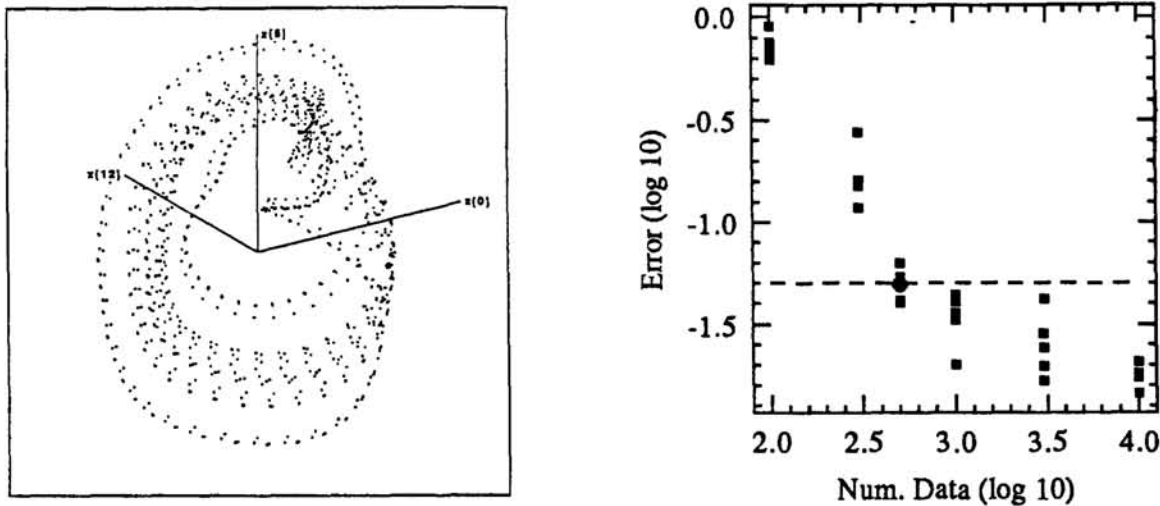

Figure 2: (A) Imbedding in three dimensions of 1000 successive points of the Mackey-Glass chaotic timeseries with delay parameter $\tau = 17$ and sampling parameter $\Delta = 6$. (B) Normalized Prediction Error vs. Number of Training Data. Squares are runs with the multi-resolution hierarchy runs. The circle is the back propagation benchmark. The horizontal line is included for visual reference only and is not intended to imply a scaling law for back propagation.

The solid lines in figure [3] show the resulting timeseries for $\tau = 17$, $a = 0.2$, and $b = 0.1$; note that it is cyclic, but not periodic. The characteristic time of the series, given by the inverse of the mean of the power spectrum, is $t_{char} \approx$ 50. Classical techniques like linear predictive coding and Gabor-Volterra-Wiener polynomial expansions typically do no better than chance when predicting beyond $t_{char}$ [10].

For purposes of comparison, the sampling parameter and prediction time are chosen to be $\Delta = 6$ and $T = 85 > t_{char}$ respectively. Figure [2A] shows a projection of the four dimensional state space imbedding onto three dimensions. The orbits of the series lie on a fuzzy two dimensional subspace which is a strange attractor of fractal dimension 2.1.

This problem has been studied by both conventional data analysis techniques and by neural network methods.

It was first studied by Farmer and Sidorowich who locally fitted linear and quadratic surfaces directly to the data. [11,10]. The exemplars in the imbedding space were stored in a $k$-$d$ tree structure in order to allow rapid determination of proximity relationships [3,4,19]. The local surface fitting method is extremely efficient computationally. This kind of approach has found wide application in the statistics community [5]. Casdagli has applied the method of radial basis functions, which is an exact interpolation method and also depends on explicit storage of the data. [9]. The radial basis functions method is a global method and becomes com-

putationally expensive when the number of exemplars is large, growing as $O(P^3)$. Both approaches yield excellent results when used as off-line algorithms, but do not seem to be well suited to real-time application domains.

For real-time applications, little *a priori* knowledge about the data can be assumed, large amounts of past data can't be stored, the function being learned may vary with time, and computing speed is essential.

Three different neural network techniques have been applied to the timeseries prediction problem, back propagation [13], self-organized, locally-tuned processing units [18,17], and an approach based on the GMDH method and simulated annealing [21]. The first two approaches can in principle be applied in real time, because they don't require explicit storage of past data and can adapt continuously. Back propagation yields better predictions since it is completely supervised, but the locally-tuned processing units learn substantially faster. The GMDH approach yields excellent results, but is computationally intensive and is probably limited to off-line use.

The multi-resolution hierarchy is intended to offer speed, precision, and the ability to adapt continuously in real time. Its application to the Mackey-Glass prediction problem is demonstrated in two different modes of operation: off-line learning and real-time learning.

## 3.1   Off-Line Learning

In off-line mode, a five level hierarchy was trained to predict the future values. At each level, a regular rectangular lattice was used, with each lattice having $A$ intervals and therefore $A + 1$ nodes per dimension. The lattice resolutions were chosen to be ($A_1 = 4$, $A_2 = 8$, $A_3 = 16$, $A_4 = 32$, $A_5 = 64$). The corresponding number of vertices in each of the virtual 4-dimensional lattices was therefore ($M_1 = 625$, $M_2 = 6,561$, $M_3 = 83,521$, $M_4 = 1,185,921$, $M_5 = 17,850,625$). The corresponding lookup table sizes were ($T_1 = 625$, $T_2 = 4096$, $T_3 = 4096$, $T_4 = 4096$, $T_5 = 4096$). Note that $T_1 = M_1$, so hashing was not required for the first layer. For all other layers, $T_\lambda < M_\lambda$, so hashing was used. For layers 3, 4, and 5, $T_\lambda \ll M_\lambda$, so hashing resulted in a dramatic reduction in the memory required. The neighborhood response function $R^\beta(\vec{x})$ was a B-spline with support in the 16 cells adjacent to each lattice point $\vec{x}^\beta$. Hash table collisions were not resolved.

The learning method used was simple gradient descent. The lookup table values were updated after the presentation of each exemplar. At each level, the training set was presented repeatedly until a convergence criterion was satisfied. The levels were trained sequentially: level 1 was trained until it converged, followed by level 2, and so on.

The performance of the system as a function of training set size is shown in figure [2B]. The normalized error is defined as [rms error]/[$\sigma$], where $\sigma$ is the standard deviation of the timeseries. For each run, a different segment of the timeseries was used. In all cases, the performance was measured on an independent test sequence consisting of the 500 exemplars immediately following the training sequence. The prediction error initially drops rapidly as the number of training data are increased,

but then begins to level out. This leveling out is most likely caused by collision noise in the hash tables. Collision resolution techniques should improve the results, but have not yet been implemented.

For training sets with 500 exemplars, the multi-resolution hierarchy achieved prediction accuracy equivalent to that of a back propagation network trained by Lapedes and Farber [13]. Their network had four linear inputs, one linear output, and two internal layers, each containing 20 sigmoidal units. The layers were fully connected yielding 541 adjustable parameters (weights and thresholds) total. They trained their network in off-line mode using conjugate gradient, which they found to be significantly faster than gradient descent.

The multi-resolution hierarchy converged in about 3.5 minutes on a Sun 3/60 for the 500 exemplar runs. Lapedes estimates that the back propagation network required probably 5 to 10 minutes of Cray X/MP time running at about 90 Mflops [12]. This would correspond to about $4,000$ to $8,000$ minutes of Sun 3/60 time. Hence, the multi-resolution hierarchy converged about three orders of magnitude faster that the back propagation network. This comparison should not be taken to be universal, since many implementations of both back propagation and the multi-resolution hierarchy are possible. Other comparisons could easily vary by factors of ten or more.

It is interesting to note that the training time for the multi-resolution hierarchy increased sub-linearly with training set size. This is because the lookup table values were varied after the presentation of each exemplar, not after presentation of the whole set. A similar effect should be observable in back propagation nets. In fact, training after the presentation of each exemplar could very likely increase the overall rate of convergence for a back propagation net.

## 3.2    Real-Time Learning

Unlike most standard curve and surface fitting methods, the multi-resolution hierarchy is extremely well-suited for real-time applications. Indeed, the standard CMAC model has been applied to the real-time control of robots with encouraging success [16,15].

Figure [3] illustrates a two level hierarchy (with 5 and 9 nodes per dimension) learning to predict the timeseries for $T = 50$ from an initial *tabula rasa* configuration (all lookup table values set to zero). The solid line is the actual timeseries data, while the dashed line are the predicted values. The predicted values lead the actual values in the graphs. Notice that the system discovers the intrinsically cyclic nature of the series almost immediately. At the end of a single pass through $9,900$ exemplars, the normalized prediction error is below 5% and the fit looks very good to the eye.

On a Sun 3/50, the algorithm required 1.4 *msec* per level to respond to and learn from each exemplar. At this rate, the two level system was able to process 360 exemplars (over 7 cycles of the timeseries) per second. This rate would be considered phenomenal for a typical back propagation network running on a Sun 3/50.

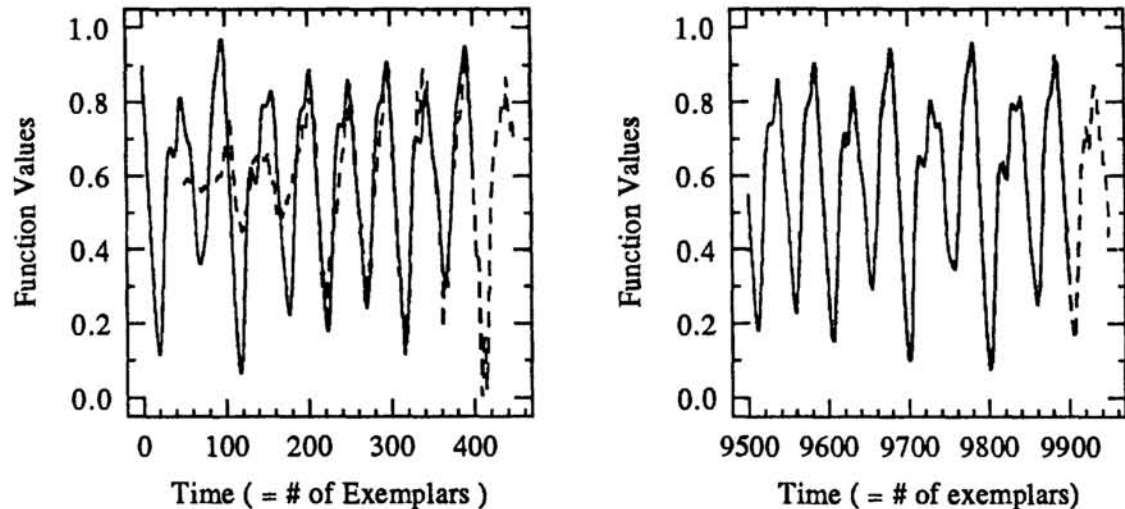

Figure 3: An example of learning to predict the Mackey-Glass chaotic timeseries in real time with a two-stage multi-resolution hierarchy.

## 4    Discussion

There are two reasons that the multi-resolution hierarchy learns much more quickly than back propagation. The first is that the hierarchy uses local representations of the input space and thus requires evaluation and modification of only a few lookup table values for each exemplar. In contrast, the complete back propagation net must be evaluated and modified for each exemplar. Second, the learning in the multi-resolution hierarchy is cast as a purely quadratic optimization procedure. In contrast, the back propagation procedure is non-linear and is plagued with a multitude of local minima and plateaus which can significantly retard the learning process.

In these respects, the multi-resolution hierarchy is very similar to the local surface fitting techniques exploited by Farmer and Sidorowich. The primary difference, however, is that the hierarchy, with its multi-resolution architecture and hash table data structures offers the flexibility needed for real time problem domains and does not require the explicit storage of past data or the creation of data structures which depend on the distribution of data.

## Acknowledgements

I gratefully acknowledge helpful comments from Chris Darken, Doyne Farmer, Alan Lapedes, Tom Miller, Terry Sejnowski, and John Sidorowich. I am especially grateful for support from ONR grant N00014-86-K-0310, AFOSR grant F49620-88-C0025, and a Purdue Army subcontract.

# References

[1] J.S. Albus. *Brain, Behavior and Robotics*. Byte Books, 1981.

[2] J.S. Albus. A new approach to manipulator control: the cerebellar model articulation controller (CMAC). *J. Dyn. Sys. Meas., Contr.*, 97:220, 1975.

[3] Jon L. Bentley. Multidimensional binary search trees in database applications. *IEEE Trans. on Software Engineering*, SE-5:333, 1979.

[4] Jon L. Bentley. Multidimensional divide and conquer. *Communications of the ACM*, 23:214, 1980.

[5] L. Breiman, J.H. Friedman, R.A. Olshen, and C.J. Stone. *Classification and Regression Trees*. Wadsworth, Monterey, CA, 1984.

[6] Peter J. Burt and Edward H. Adelson. The laplacian pyramid as a compact image code. *IEEE Trans. Communications*, COM-31:532, 1983.

[7] Peter J. Burt and Edward H. Adelson. A multiresolution spline with application to image mosaics. *ACM Trans. on Graphics*, 2:217, 1983.

[8] J.L. Carter and M.N. Wegman. Universal classes of hash functions. In *Proceedings of the Ninth Annual SIGACT Conference*, 1977.

[9] M. Casdagli. *Nonlinear Prediction of Chaotic Time Series*. Technical Report, Queen Mary College, London, 1988.

[10] J.D. Farmer and J.J. Sidorowich. *Exploiting Chaos to Predict the Future and Reduce Noise*. Technical Report, Los Alamos National Laboratory, Los Alamos, New Mexico, 1988.

[11] J.D. Farmer and J.J. Sidorowich. Predicting chaotic time series. *Physical Review Letters*, 59:845, 1987.

[12] A. Lapedes. 1988. Personal communication.

[13] A.S. Lapedes and R. Farber. *Nonlinear Signal Processing Using Neural Networks: Prediction and System Modeling*. Technical Report, Los Alamos National Laboratory, Los Alamos, New Mexico, 1987.

[14] M.C. Mackey and L. Glass. Oscillation and chaos in physiological control systems. *Science*, 197:287.

[15] W. T. Miller, F. H. Glanz, and L. G. Kraft. Application of a general learning algorithm to the control of robotic manipulators. *International Journal of Robotics Research*, 6(2):84, 1987.

[16] W. Thomas Miller. Sensor-based control of robotic manipulators using a general learning algorithm. *IEEE Journal of Robotics and Automation*, RA-3(2):157, 1987.

[17] J. Moody and C. Darken. Fast learning in networks of locally-tuned processing units. *Neural Computation*, 1989. To Appear.

[18] J. Moody and C. Darken. Learning with localized receptive fields. In Touretzky, Hinton, and Sejnowski, editors, *Proceedings of the 1988 Connectionist Models Summer School*, Morgan Kaufmann, Publishers, 1988.

[19] S. Omohundro. Efficient algorithms with neural network behavior. *Complex Systems*, 1:273.

[20] T. Sejnowski and C. Rosenberg. Parallel networks that learn to pronounce English text. *Complex Systems*, 1:145, 1987.

[21] M.F. Tenorio and W.T. Lee. Self-organized neural networks for the identification problem. Poster paper presented at the Neural Information Processing Systems Conference, 1988.